# A general framework for investigating how far the decoding process in the brain can be simplified

**Masafumi Oizumi**[1]**, Toshiyuki Ishii**[2]**, Kazuya Ishibashi**[1]
Toshihiko Hosoya[2], Masato Okada[1,2]
oizumi@mns.k.u-tokyo.ac.jp
tishii@brain.riken.jp,kazuya@mns.k.u-tokyo.ac.jp
hosoya@brain.riken.jp, okada@k.u-tokyo.ac.jp
[1] University of Tokyo, Kashiwa-shi, Chiba, JAPAN
[2] RIKEN Brain Science Institute, Wako-shi, Saitama, JAPAN

## Abstract

"How is information decoded in the brain?" is one of the most difficult and important questions in neuroscience. Whether neural correlation is important or not in decoding neural activities is of special interest. We have developed a general framework for investigating how far the decoding process in the brain can be simplified. First, we hierarchically construct simplified probabilistic models of neural responses that ignore more than $K$th-order correlations by using a maximum entropy principle. Then, we compute how much information is lost when information is decoded using the simplified models, i.e., "mismatched decoders". We introduce an information theoretically correct quantity for evaluating the information obtained by mismatched decoders. We applied our proposed framework to spike data for vertebrate retina. We used 100-ms natural movies as stimuli and computed the information contained in neural activities about these movies. We found that the information loss is negligibly small in population activities of ganglion cells even if all orders of correlation are ignored in decoding. We also found that if we assume stationarity for long durations in the information analysis of dynamically changing stimuli like natural movies, pseudo correlations seem to carry a large portion of the information.

## 1 Introduction

An ultimate goal of neuroscience is to elucidate how information is encoded and decoded by neural activities. To investigate what information is encoded by neurons in certain area of the brain, the mutual information between stimuli and neural responses is often calculated. In the analysis of mutual information, it is implicitly assumed that encoded information is decoded by an optimal decoder, which exactly matches the encoder. In other words, the brain is assumed to have full knowledge of the encoding process. Generally, if the neural activities are correlated, the amount of data needed for the optimal decoding scales exponentially with the number of neurons. Since a large amount of data and many complex computations are needed for optimal decoding, the assumption of an optimal decoder in the brain is doubtful.

The reason mutual information is widely used in neuroscience despite the doubtfulness of the optimal decoder is that we are completely ignorant of how information is decoded in the brain. Thus, we simply evaluate the maximal amount of information that can be extracted from neural activities by calculating the mutual information. To address this lack of knowledge, we can ask a different question: "How much information can be obtained by a decoder that has partial knowledge of the encoding process?" [10, 14] We call this type of a decoder "simplified decoder" or a "mismatched decoder". For example, an independent decoder is a simplified decoder; it takes only the marginal

distribution of the neural responses into consideration and ignores the correlations between neuronal activities. The independent decoder is of particular importance because several studies have shown that maximum likelihood estimation can be implemented by a biologically plausible network [2, 4]. If it is experimentally shown that a sufficiently large portion of information is obtained by the independent decoder, we can say that the brain may function in a manner similar to the independent decoder. In this context, Nirenberg *et al.* computed the amount of information obtained by the independent decoder in pairs of retinal ganglion cells activities [10]. They showed that no pair of cells showed a loss of information greater than 11%. Because only pairs of cells were considered in their analysis, it has not been still elucidated whether correlations are not important in population activities.

To elucidate whether correlations are important or not in population activities, we have developed a general framework for investigating the importance of correlation in decoding neural activities. When population activities are analyzed, we have to deal with not only second-order correlations but also higher-order correlations in general. Therefore, we need to hierarchically construct simplified decoders that account of up to $K$th-order correlations, where $K = 1, 2, ..., N$. By computing how much information is obtained by the simplified decoders, we investigate how many orders of correlation should be taken into account to extract enough information. To compute the information obtained by the mismatched decoders, we introduce a information theoretically correct quantity derived by Merhav *et al.* [8]. Information for mismatched decoders previously proposed by Nirenberg and Latham is the lower bound on the correct information [5, 11]. Because this lower bound can be very loose and their proposed information can be negative when many cells are analyzed as is shown in the paper, we need to accurately evaluate the information obtained by mismatched decoders.

The plan of the paper is as follows. In Section 2, we describe a way of computing the information that can be extracted from neural activities by mismatched decoders using the information derived by Merhav *et al.*. Using analytical computation, we demonstrate how information for mismatched decoders previously proposed by Nirenberg and Latham differs from the correct information derived by Merhav *et al.*, especially when many cells are analyzed. In Section 3, we apply our framework to spike data for ganglion cells in the salamander retina. We first describe the method of hierarchically constructing simplified decoders by using the maximum entropy principle [12]. We then compute the information obtained with the simplified decoders. We find that more than 90% of the information can be extracted from the population activities of ganglion cells even if all orders of correlations are ignored in decoding. We also describe the problem of previous studies [10, 12] in which the stationarity of stimuli is assumed for a duration that is too long. Using a toy model, we demonstrate that pseudo correlations seem to carry a large portion of the information because of the stationarity assumption.

## 2   Information for mismatched decoders

Let us consider how much information about stimuli can be extracted from neural responses. We assume that we experimentally obtain the conditional probability distribution $p(\mathbf{r}|s)$ that neural responses $\mathbf{r}$ are evoked by stimulus $s$. We can say that the stimulus is encoded by neural response $\mathbf{r}$, which obeys the distribution $p(\mathbf{r}|s)$. We call $p(\mathbf{r}|s)$ the "encoding model". The maximal amount of information obtained with the optimal decoder can be evaluated by using the mutual information:

$$I = - \int d\mathbf{r} p(\mathbf{r}) \log_2 p(\mathbf{r}) + \int d\mathbf{r} \sum_s p(s)p(\mathbf{r}|s) \log_2 p(\mathbf{r}|s), \tag{1}$$

where $p(\mathbf{r}) = \sum_s p(\mathbf{r}|s)p(s)$ and $p(s)$ is the prior probability of stimuli. In the optimal decoder, the probability distribution $q(\mathbf{r}|s)$ that exactly matches the encoding model $p(\mathbf{r}|s)$ is used for decoding; that is, $q(\mathbf{r}|s) = p(\mathbf{r}|s)$. We call $q(\mathbf{r}|s)$ the "decoding model". We can also compute the maximal amount of information obtained by a decoder using a decoding model $q(\mathbf{r}|s)$ that does not match the encoding model $p(\mathbf{r}|s)$ by using an equation derived by Merhav *et al.* [8]:

$$I^*(\beta) = - \int d\mathbf{r} p(\mathbf{r}) \log_2 \sum_s p(s)q(\mathbf{r}|s)^\beta + \int d\mathbf{r} \sum_s p(s)p(\mathbf{r}|s) \log_2 q(\mathbf{r}|s)^\beta, \tag{2}$$

where $\beta$ takes the value that maximizes $I^*(\beta)$. Thus, $\beta$ is the value that satisfies $\partial I^*/\partial \beta = 0$. We call a decoder using the mismatched decoding model a "mismatched decoder".

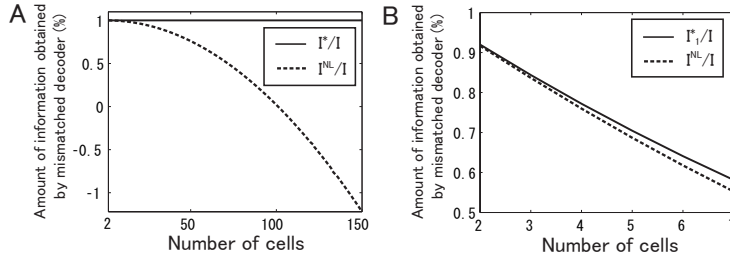

Figure 1: Comparison between correct information $I^*$ derived by Merhav *et al.* and Nirenberg-Latham information $I^{NL}$. A: Difference between $I^*/I$ (solid line) and $I^{NL}/I$ (dotted line) in Gaussian model where correlations and derivatives of mean firing rates are uniform. Correlation parameter $c = 0.01$. B: Difference between $I_1^*/I$ (solid line) and $I_1^{NL}/I$ (dotted line) when spike data in Figure 3A are used. For this spike data and other spike data analyzed, Nirenberg-Latham information provides a tight lower bound on the correct information, possibly because the number of cells is small.

Previously, Nirenberg and Latham proposed that the information obtained by mismatched decoders can be evaluated by using [11]

$$I^{NL} = - \int d\mathbf{r} p(\mathbf{r}) \log_2 \sum_s p(s) q(\mathbf{r}|s) + \int d\mathbf{r} \sum_s p(s) p(\mathbf{r}|s) \log_2 q(\mathbf{r}|s). \qquad (3)$$

We call their proposed information "Nirenberg-Latham information". If we set $\beta = 1$ in Eq. 2, we obtain Nirenberg-Latham information, $I^*(1) = I^{NL}$. Thus, Nirenberg-Latham information does not give correct information; instead, it simply provides the lower bound on the correct information, $I^*(\beta)$, which is the maximum value with respect to $\beta$ [5, 8]. The lower bound provided by Nirenberg-Latham information can be very loose and the Nirenberg-Latham information can be negative when many cells are analyzed.

**Theoretical evaluation of information $I$, $I^*$, and $I^{NL}$**

We consider the problem where mutual information is computed when stimulus $s$, which is a single variable, and slightly different stimulus $s + \Delta s$ are presented. We assume the prior probability of stimuli, $p(s)$ and $p(s + \Delta s)$, are equal: $p(s) = p(s + \Delta s) = 1/2$. Neural responses evoked by the stimuli are denoted by $\mathbf{r}$, which is considered here to be the neuron firing rate. When the difference between two stimuli is small, the conditional probability $p(\mathbf{r}|s + \Delta s)$ can be expanded with respect to $\Delta s$ as $p(\mathbf{r}|s+\Delta s) = p(\mathbf{r}|s) + p'(\mathbf{r}|s)\Delta s + \frac{1}{2}p''(\mathbf{r}|s)(\Delta s)^2 + ...$, where $'$ represents differentiation with respect to $s$. Using the expansion, to leading order of $\Delta s$, we can write mutual information $I$ as

$$I = \frac{\Delta s^2}{8} \int d\mathbf{r} \frac{(p'(\mathbf{r}|s))^2}{p(\mathbf{r}|s)}, \qquad (4)$$

where $\int d\mathbf{r} \frac{p'(\mathbf{r}|s)^2}{p(\mathbf{r}|s)}$ is the Fisher information. Thus, we can see that the mutual information is proportional to the Fisher information when $\Delta s$ is small. Similarly, the correct information $I^*$ for the mismatched decoders and the Nirenberg-Latham information $I^{NL}$ can be written as

$$I^* = \frac{\Delta s^2}{8} \left( \int d\mathbf{r} \frac{p'(\mathbf{r}|s) q'(\mathbf{r}|s)}{q(\mathbf{r}|s)} \right)^2 \left( \int d\mathbf{r} \frac{p(\mathbf{r}|s)(q'(\mathbf{r}|s))^2}{q(\mathbf{r}|s)^2} \right)^{-1}, \qquad (5)$$

$$I^{NL} = \frac{\Delta s^2}{8} \left( - \int d\mathbf{r} p(\mathbf{r}|s) \left( \frac{q'(\mathbf{r}|s)}{q(\mathbf{r}|s)} \right)^2 + 2 \int d\mathbf{r} \frac{p'(\mathbf{r}|s) q(\mathbf{r}|s)}{q(\mathbf{r}|s)} \right). \qquad (6)$$

Taking into consideration the proportionality of the mutual information to the Fisher information, we can interpret that $\left( \int d\mathbf{r} \frac{p'(\mathbf{r}|s)q'(\mathbf{r}|s)}{q(\mathbf{r}|s)} \right)^2 \left( \int d\mathbf{r} \frac{p(\mathbf{r}|s)(q'(\mathbf{r}|s))^2}{q(\mathbf{r}|s)^2} \right)^{-1}$ in Eq. 5 is a Fisher information-like quantity for mismatched decoders.

Let us consider the case in which the encoding model $p(\mathbf{r}|s)$ obeys the Gaussian distribution

$$p(\mathbf{r}|s) = \frac{1}{Z} \exp\left(-\frac{1}{2}(\mathbf{r} - \mathbf{f}(s))^T \mathbf{C}^{-1}(\mathbf{r} - \mathbf{f}(s))\right), \tag{7}$$

where $^T$ stands for the transpose operation, $\mathbf{f}(s)$ is the mean firing rates given stimulus $s$, and $\mathbf{C}$ is the covariance matrix. We consider an independent decoding model $q(\mathbf{r}|s)$ that ignores correlations:

$$q(\mathbf{r}|s) = \frac{1}{Z_D} \exp\left(-\frac{1}{2}(\mathbf{r} - \mathbf{f}(s))^T \mathbf{C}_D^{-1}(\mathbf{r} - \mathbf{f}(s))\right), \tag{8}$$

where $\mathbf{C}_D$ is the diagonal covariance matrix obtained by setting the off-diagonal elements of $\mathbf{C}$ to 0. If the Gaussian integral is performed for Eqs. 4-5, $I$, $I^*$, and $I^{NL}$ can be written as

$$I = \frac{\Delta s^2}{8} \mathbf{f}'^T(s) \mathbf{C}^{-1} \mathbf{f}'(s), \tag{9}$$

$$I^* = \frac{\Delta s^2}{8} \frac{(\mathbf{f}'^T(s) \mathbf{C}_D^{-1} \mathbf{f}'(s))^2}{\mathbf{f}'^T(s) \mathbf{C}_D^{-1} \mathbf{C} \mathbf{C}_D^{-1} \mathbf{f}'(s)}, \tag{10}$$

$$I^{NL} = \frac{\Delta s^2}{8} \left(-\mathbf{f}'^T(s) \mathbf{C}_D^{-1} \mathbf{C} \mathbf{C}_D^{-1} \mathbf{f}'(s) + 2\mathbf{f}'^T(s) \mathbf{C}_D^{-1} \mathbf{f}'(s)\right). \tag{11}$$

The correct information obtained by the independent decoder for the Gaussian model (Eq. 10) is inversely proportional to the decoding error of $s$ when the independent decoder is applied, which was computed from the generalized Cramér Rao bound by Wu $et$ $al.$ [14].

As a simple example, we consider a uniform correlation model [1, 14] in which covariance matrix $\mathbf{C}$ is given by $C_{ij} = \sigma^2[\delta_{ij} + c(1 - \delta_{ij})]$ and assume that the derivatives of the firing rates are uniform: that is $f_i' = f'$. In this case, $I$, $I^*$, and $I^{NL}$ can be computed using

$$I = \frac{\Delta s^2}{8} \frac{Nf'^2}{\sigma^2(Nc + 1 - c)}, \tag{12}$$

$$I^* = \frac{\Delta s^2}{8} \frac{Nf'^2}{\sigma^2(Nc + 1 - c)}, \tag{13}$$

$$I^{NL} = \frac{\Delta s^2}{8} \frac{(-c(N - 1) + 1)Nf'^2}{\sigma^2}, \tag{14}$$

where $N$ is the number of cells. We can see that $I^*$ is equal to $I$, which means that information is not lost even if correlation is ignored in the decoding process. Figure 1A shows $I^{NL}/I$ and $I^*/I$ when the degree of correlation $c$ is 0.01. As shown in Figure 1A, the difference between the correct information $I^*$ and Nirenberg-Latham information $I^{NL}$ is very large when the number of cells $N$ is large. When $N > \frac{c+1}{c}$, $I^{NL}$ is negative. Analysis showed that using Nirenberg-Latham information $I^{NL}$ as a lower bound on the correct information $I^*$ can lead to wrong conclusions, especially when many cells are analyzed.

## 3 Analysis of information in population activities of ganglion cells

### 3.1 Methods

We analyzed the data obtained when $N = 7$ retinal ganglion cells were simultaneously recorded using a multielectrode array. The stimulus was a natural movie, which was 200 s long and repeated 45 times. We divided the movie into many short natural movies and considered them as stimuli over which information contained in neural activities is computed. For instance, when it was divided into 10-s-long natural movies, there were 20 stimuli. Figure 2A shows the response of the seven retinal ganglion cells to natural movies from 0 to 10 s in length. To apply information theoretic techniques, we first discretized the time into small time bins $\Delta\tau$ and indicated whether a spike was emitted or not in each time bin with a binary variable: $\sigma_i = 1$ means that the cell $i$ spiked and $\sigma_i = 0$ means that it did not spike. We set the length of the time, $\Delta\tau$, to 5 ms so that it was short enough to avoid two spikes falling into the same bin. In this way, the spike pattern of ganglion cells was transformed into an $N$-letter binary word, $\boldsymbol{\sigma} = \{\sigma_1, \sigma_2, ..., \sigma_N\}$, as shown in Figure 2B. Then, we determined the

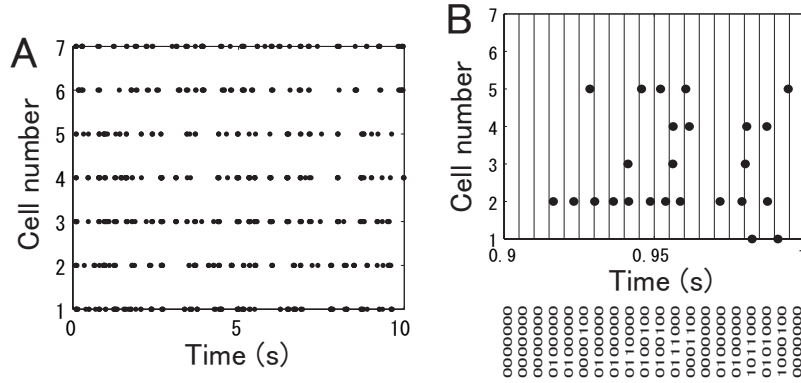

Figure 2: A: Raster plot of seven retinal ganglion cells responding to a natural movie. B: Transformation of spike trains into binary words.

frequency with which a particular spike pattern, $\boldsymbol{\sigma}$, was observed during each stimulus and estimated the conditional probability distribution $p_{\text{data}}(\boldsymbol{\sigma}|s)$ from experimental data. Using these conditional probabilities, we evaluated the information contained in $N$-letter binary words $\boldsymbol{\sigma}$.

Generally, the joint probability of $N$ binary variables can be written as [9]

$$p_N(\boldsymbol{\sigma}) = \frac{1}{Z} \exp\left[\sum_i \theta_i \sigma_i + \sum_{i<j} \theta_{ij} \sigma_i \sigma_j + \cdots + \theta_{12\ldots N} \sigma_1 \sigma_2 \ldots \sigma_N\right]. \tag{15}$$

This type of probability distribution is called a log-linear model. Because the number of parameters in a log-linear model is equal to the number of all possible configurations of an $N$-letter binary word $\boldsymbol{\sigma}$, we can determine the values of parameters so that the log-linear model $p_N(\boldsymbol{\sigma})$ exactly matches empirical probability distribution $p_{\text{data}}(\boldsymbol{\sigma})$: that is, $p_N(\boldsymbol{\sigma}) = p_{\text{data}}(\boldsymbol{\sigma})$.

To compute the information for mismatched decoders, we construct simplified models of neural responses that partially match the empirical distribution, $p_{\text{data}}(\boldsymbol{\sigma})$. The simplest model is an "independent model" $p_1(\boldsymbol{\sigma})$, where only the average of each $\sigma_i$ agrees with the experimental data: that is, $\langle \sigma_i \rangle_{p_1(\boldsymbol{\sigma})} = \langle \sigma_i \rangle_{p_{\text{data}}(\boldsymbol{\sigma})}$. There are many possible probability distributions that satisfy these constraints. In accordance with the maximum entropy principle [12], we choose the one that maximizes entropy $H$, $H = -\sum_{\boldsymbol{\sigma}} p_1(\boldsymbol{\sigma}) \log p_1(\boldsymbol{\sigma})$. The resulting maximum entropy distribution is

$$p_1(\boldsymbol{\sigma}) = \frac{1}{Z_1} \exp\left[\sum_i \theta_i^{(1)} \sigma_i\right]. \tag{16}$$

in which model parameters $\boldsymbol{\theta}^{(1)}$ are determined so that the constraints are satisfied. This model corresponds to a log-linear model in which all orders of correlation parameters $\{\theta_{ij}, \theta_{ijk}, \ldots, \theta_{12\ldots N}\}$ are omitted. If we perform maximum likelihood estimation of model parameters $\boldsymbol{\theta}^{(1)}$ in the log-linear model, the result is that the average $\sigma_i$ under the log-linear model equals the average $\sigma_i$ found in the data: that is, $\langle \sigma_i \rangle_{p_1(\boldsymbol{\sigma})} = \langle \sigma_i \rangle_{p_{\text{data}}(\boldsymbol{\sigma})}$. This result is identical to the constraints of the maximum entropy model. Generally, the maximum entropy method is equivalent to maximum likelihood fitting of a log-linear model [6].

Similarly, we can consider a "second-order correlation model" $p_2(\boldsymbol{\sigma})$, which is consistent with not only the averages of $\sigma_i$ but also the averages of all products $\sigma_i \sigma_j$ found in the data. Maximizing the entropy with constraints $\langle \sigma_i \rangle_{p_2(\boldsymbol{\sigma})} = \langle \sigma_i \rangle_{p_{\text{data}}(\boldsymbol{\sigma})}$ and $\langle \sigma_i \sigma_j \rangle_{p_2(\boldsymbol{\sigma})} = \langle \sigma_i \sigma_j \rangle_{p_{\text{data}}(\boldsymbol{\sigma})}$, we obtain

$$p_2(\boldsymbol{\sigma}) = \frac{1}{Z_2} \exp\left[\sum_i \theta_i^{(2)} \sigma_i + \sum_{i,j} \theta_{ij}^{(2)} \sigma_i \sigma_j\right], \tag{17}$$

in which model parameters $\boldsymbol{\theta}^{(2)}$ are determined so that the constraints are satisfied. The procedure described above can also be used to construct a "$K$th-order correlation model" $p_K(\boldsymbol{\sigma})$. If we substitute the simplified models of neural responses $p_K(\boldsymbol{\sigma}|s)$ into mismatched decoding models $q(\boldsymbol{\sigma}|s)$ in

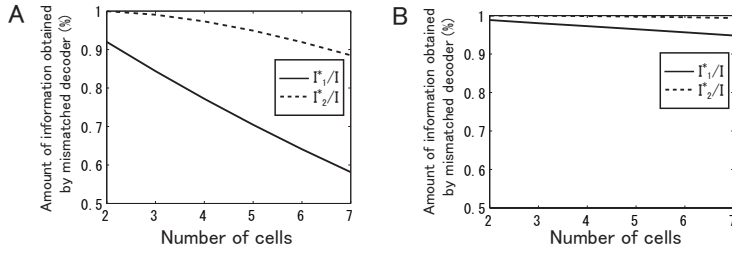

Figure 3: Dependence of amount of information obtained by simplified decoders on number of ganglion cells analyzed. Same spike data obtained from retinal ganglion cells responding to a natural movie were used to obtain analysis results shown in panels A and B. A: 10-s-long natural movie B: 100-ms-long natural movie

Eq. 2, we can compute the amount of information that can be obtained when more than $K$th-order correlations are ignored in the decoding,

$$I_K^*(\beta) = -\sum_{\boldsymbol{\sigma}} p_N(\boldsymbol{\sigma}) \log_2 \sum_s p(s) p_K(\boldsymbol{\sigma}|s)^\beta + \sum_s p(s) \sum_{\boldsymbol{\sigma}} p_N(\boldsymbol{\sigma}|s) \log_2 p_K(\boldsymbol{\sigma}|s)^\beta. \quad (18)$$

By evaluating the ratio of information, $I_K^*/I$, we can infer how many orders of correlation should be taken into account to extract enough information.

## 3.2 Results

First, we investigated how the ratio of information obtained by an independent model, $I_1^*/I$, and that obtained by a second-order correlation model, $I_2^*/I$, changed when the number of cells analyzed was changed. We set the length of the stimulus to 10 s. We could obtain 20 kinds of stimuli from a 200-s-long natural movie (see Methods). In previous studies, comparable length stimuli (7 s for Nirenberg *et al.*'s study [10] and 20 s for Schneidman *et al.*'s study [12]) were used. When two neurons were analyzed, there were 21 possible combinations for choosing 2 cells out of 7 cells, which is the total number of cells simultaneously recorded. We computed the average value of $I_K^*/I$ for $K = 1, 2$ over all possible combinations of cells. Figure 3A shows that $I_1^*/I$ and $I_2^*/I$ monotonically decreased when the number of cells was increased. A comparison between the correct information, $I_1^*/I$, and Nirenberg-Latham information, $I_1^{NL}/I$ where $I_1^{NL} = I_1^*(\beta = 1)$, is shown in Figure 1B. When only two cells were considered, $I_1^*/I$ exceeded 90%, which means that ignoring correlation leads to only a small loss of information. This is consistent with the result obtained by Nirenberg *et al.* [10]. However, when all cells ($N = 7$) were used in the analysis, $I_1^*/I$ becomes only about 60%. Thus, correlation seems to be much more important for decoding when population activities are considered than when only two cells are considered. At least, we can say that qualitatively different things occur when large populations of cells are analyzed, as Schneidman *et al.* pointed out [12].

We should be careful about concluding from the results shown in Figure 3A that correlation is important for decoding. In this analysis, we considered a 10-s-long stimuli and assumed stationarity during each stimulus. By stationarity we mean that we assumed spikes are generated by a single process that can be described by a single conditional distribution $p(\boldsymbol{\sigma}|s)$. Because the natural movies change much more rapidly and our visual system has much higher time resolution than 10 s [13], we also considered shorter stimuli. In Figure 3B, we computed $I_1^*/I$ and $I_2^*/I$ over 100-ms-long natural movies. In this case, we could obtain 2000 stimuli from the 200-s-long natural movie. When the length of each stimulus was 100 ms, no spikes occurred while some stimuli were presented. We removed those stimuli and used the remaining stimuli for the analysis. In this case, the amount of information obtained by independent model $I_1^*$ was more than 90% even when all cells ($N = 7$) were considered. Although 100 ms may still be too long to be considered as a single process, the result shown in Figure 3B reflects a situation that our brain has to deal with, that is more realistic than that reflected in Figure 3A. Figure 4A shows the dependence of information obtained by simplified decoders on the length of stimulus. In this analysis, we changed the length of the stimulus from 100 ms to 10 s and computed $I_1^*/I$ and $I_2^*/I$ for activities of $N = 7$ cells. We also analyzed additional experimental data obtained when $N = 6$ retinal ganglion cells were simultaneously recorded from

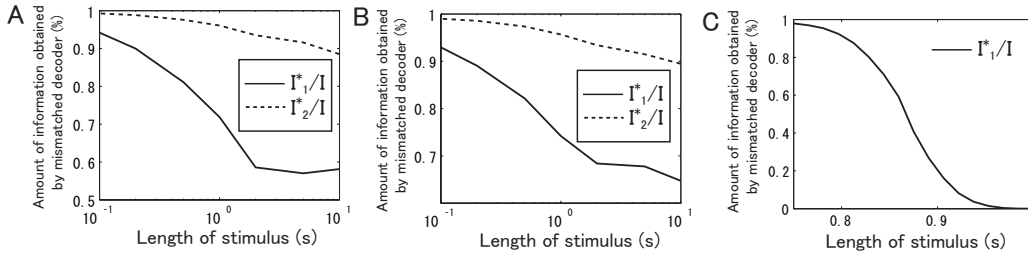

Figure 4: Dependence of amount of information obtained by simplified decoders on length of stimuli. Stimulus was same natural movie for both panels, but spike data obtained from retinas of different salamander were used in panels A and B. A: Seven simultaneously recorded ganglion cells B: Six simultaneously recorded ganglion cells C: Artificial spike data generated according to the firing rates shown in Figure 5A

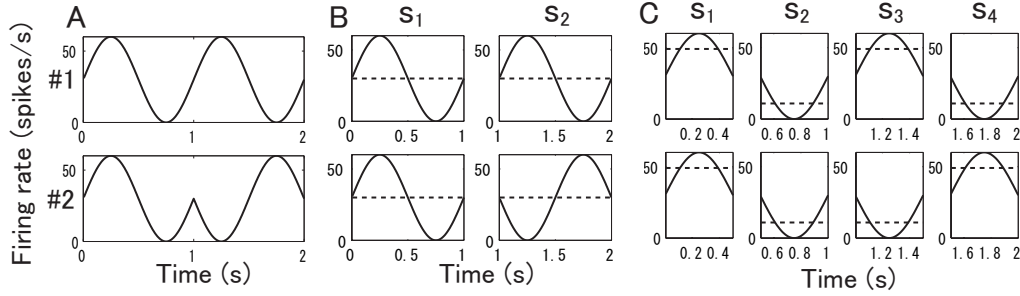

Figure 5: Firing rates of two model cells. Rate of cell #1 shown in top panel; rate of cell #2 is shown in bottom panel. A: Firing rates from 0 to 2 s. B: Firing rates (solid line) and mean firing rates (dashed line) when stimulus was 1 s long. C: Firing rates (solid line) and mean firing rates (dashed line) when stimulus was 500 ms long.

another salamander retina. The same 200-s-long natural movie was used as a stimulus for Figure 4B as for Figure 4A, and the activities of $N = 6$ cells were analyzed. Figure 4B shows the result. We can clearly see the same tendency as shown in Figures 4A and B: the amount of information decoded by the simplified decoders monotonically increased as the length of the stimulus was shortened.

To clarify the reason the correlation becomes less important as the stimulus is shortened, we used the toy model shown in Figure 5. We considered the case in which two cells fire independently in accordance with a Poisson process and performed an analysis similar to the one we did for the actual spike data. We used simulated spike data for the two cells generated in accordance with the firing rates shown in Figure 5A. The firing rates with a 2-s stimulus sinusoidally change with time. We divided the 2-s-long stimulus into two 1-s-long stimulus, $s_1$ and $s_2$, as shown in Figure 5B. Then, we computed mutual information $I$ and the information obtained by independent model $I_1^*$ over $s_1$ and $s_2$. Because the two cells fired independently, there were no correlations between two cells essentially. However, there was pseudo correlation due to the assumption of stationarity for the dynamically changing stimulus. The pseudo correlation was high for $s_1$ and low for $s_2$. This means that "correlation" plays an important role in discriminating two stimuli, $s_1$ and $s_2$. In contrast, the mean firing rates of the two cells during each stimulus were equal for $s_1$ and $s_2$. Therefore, if the stimulus is 1 s long, we cannot discriminate two stimuli by using the independent model, that is, $I_1^* = 0$. We also considered the case in which the stimulus was 0.5 s long, as shown in Figure 5C. In this case, pseudo correlations again appeared but there was a significant difference in the mean firing rates between the stimuli. Thus, the independent model can be used to extract almost all the information. The dependence of $I_1^*/I$ on the stimulus length is shown in Figure 4C. Behaviors similar to those represented in Figure 4C were also observed in the analysis of the actual spike data for retinal ganglion cells (Figure 4A and 4B). Even if we observe that correlation carries a significant large portion of information for longer stimuli compared with the speed of change in the firing rates,

it may simply be caused by meaningless pseudo correlation. To assess the role of correlation in information processing, the stimuli used should be short enough to think neural responses to these stimuli generated by a single process.

## 4  Summary and Discussion

We described a general framework for investigating how far the decoding process in the brain can be simplified. We computed the amount of information that can be extracted by using simplified decoders constructed using a maximum entropy model, i.e., mismatched decoders. We showed that more than 90% of the information encoded in retinal ganglion cells activities can be decoded by using an independent model that ignores correlation. Our results imply that the brain uses a simplified decoding strategy in which correlation is ignored.

When we computed the information obtained by the independent model, we regarded a 100-ms-long natural movie as one stimulus. However, when we considered stimuli that were long compared with the speed of the change in the firing rates as one stimulus, correlation carried a large portion of information. This is due to pseudo correlation, which is observed if stationarity is assumed for long durations. The human visual system can process visual information in less than 150 ms [13]. We should set the length of the stimulus appropriately by taking the time resolution of our visual system into account.

Our results do not imply that any kind of correlation does not carry much information because we dealt only with correlated spikes within a 5-ms time bin. In our analysis, we did not analyze the correlation on a longer time scale, which can be observed in the activities of retinal ganglion cells [7]. We also did not investigate the information carried by the relative timing of spikes [3]. Further investigations are needed for these types of correlation. Our approach of comparing the mutual information with the information obtained by simplified decoders can also be used for studying other types of correlations.

## References

[1]  Abbott, L. F., & Dayan, P. (1999). *Neural Comput.*, 11, 91-101.

[2]  Deneve, S., Latham, P. E., & Pouget, A. (1999). *Nature Neurosci.*, 2, 740-745.

[3]  Gollish, S., & Meister, M. (2008). *Science*, 319, 1108-1111.

[4]  Jazayeri, M. & Movshon, J. A. (2006). *Nature Neurosci.*, 9, 690-696.

[5]  Latham, P. E., & Nirenberg, S. (2005). *J. Neurosci.*, 25, 5195-5206.

[6]  MacKay, D. (2003). *Information Theory, Inference and Learning Algorithms* (Cambridge Univ. Press, Cambridge, England).

[7]  Meister, M., & Berry, M. J. II (1999). *Neuron*, 22, 435-450.

[8]  Merhav, N., Kaplan, G., Lapidoth, A., & Shamai Shitz, S. (1994). *IEEE Trans. Inform. Theory*, 40, 1953-1967.

[9]  Nakahara, H., & Amari, S. (2002). *Neural Comput.*, 14, 2269-2316.

[10]  Nirenberg, S., Carcieri, S. M., Jacobs, A. L., & Latham, P. E. (2001). *Nature*, 411, 698-701.

[11]  Nirenberg, S., & Latham, P. (2003). *Proc. Natl. Acad. Sci. USA*, 100, 7348-7353.

[12]  Schneidman, E., Berry, M. J. II, Segev, R., & Bialek. W. (2006). *Nature*, 440, 1007-1012.

[13]  Thorpe, S., Fize, D., & Marlot, C. (1996). *Nature*, 381, 520-522.

[14]  Wu, S., Nakahara, H., & Amari, S. (2001). *Neural Comput.*, 13, 775-797.

